# Counting Solution Clusters in Graph Coloring Problems Using Belief Propagation

**Lukas Kroc**        **Ashish Sabharwal**        **Bart Selman**
Department of Computer Science
Cornell University, Ithaca NY 14853-7501, U.S.A.
`{kroc,sabhar,selman}@cs.cornell.edu` [*]

## Abstract

We show that an important and computationally challenging solution space feature of the graph coloring problem (COL), namely the number of clusters of solutions, can be accurately estimated by a technique very similar to one for counting the number of solutions. This cluster counting approach can be naturally written in terms of a new factor graph derived from the factor graph representing the COL instance. Using a variant of the Belief Propagation inference framework, we can efficiently approximate cluster counts in random COL problems over a large range of graph densities. We illustrate the algorithm on instances with up to $100,000$ vertices. Moreover, we supply a methodology for computing the number of clusters exactly using advanced techniques from the knowledge compilation literature. This methodology scales up to several hundred variables.

## 1   Introduction

Message passing algorithms, in particular Belief Propagation (BP), have been very successful in efficiently computing interesting properties of succinctly represented large spaces, such as joint probability distributions. Recently, these techniques have also been applied to compute properties of discrete spaces, in particular, properties of the space of solutions of combinatorial problems. For example, for propositional satisfiability (SAT) and graph coloring (COL) problems, marginal probability information about the uniform distribution over solutions (or similar combinatorial objects) has been the key ingredient in the success of BP-like algorithms. Most notably, the survey propagation (SP) algorithm utilizes this information to solve very large hard random instances of these problems [3, 11].

Earlier work on random ensembles of Constraint Satisfaction Problems (CSPs) has shown that the computationally hardest instances occur near phase boundaries, where instances go from having many globally satisfying solutions to having no solution at all (a "solution-focused picture"). In recent years, this picture has been refined and it was found that a key factor in determining the hardness of instances in terms of search algorithm (or sampling algorithm) is the question: *how are the solutions spatially distributed within the search space?* This has made the structure of the solution space in terms of its clustering properties a key factor in determining the performance of combinatorial search methods (a "cluster-focused picture"). Can BP-like algorithms be used to provide such cluster-focused information? For example, how many clusters are there in a solution space? How big are the clusters? How are they organized? Answers to such questions will shed further light into our understanding of these hard combinatorial problems and lead to better algorithmic approaches for reasoning about them, be it for finding one solution or answering queries of probabilistic inference about the set of solutions. The study of the solution space geometry has indeed been the focus

---
[*]This work was supported by IISI, Cornell University (AFOSR grant FA9550-04-1-0151), DARPA (REAL grant FA8750-04-2-0216), and NSF (grant 0514429).

of a number of recent papers [e.g. 1, 2, 3, 7, 9, 11], especially by the statistical physics community, which has developed extensive theoretical tools to analyze such spaces under certain structural assumptions and large size limits. We provide a purely combinatorial method for counting the number of clusters, which is applicable even to small size problems and can be approximated very well by message passing techniques.

Solutions can be thought of as 'neighbors' if they differ in the value of one variable, and the transitive closure of the neighbor relation defines clusters in a natural manner. Counting the number of clusters is a challenging problem. To begin with, it is not even clear what is the best succinct way to represent clusters. One relatively crude but useful way is to represent a cluster by the set of 'backbone' variables in that cluster, i.e., variables that take a fixed value in all solutions within the cluster. Interestingly, while it is easy (polynomial time) to verify whether a variable assignment is indeed a solution of CSP, the same check is much harder for a candidate cluster represented by the set of its backbone variables.

We propose one of the first scalable methods for estimating the number of clusters of solutions of graph coloring problems using a belief propagation like algorithm. While the naïve method, based on enumeration of solutions and pairwise distances, scales to graph coloring problems with 50 or so nodes and a recently proposed local search based method provides estimates up to a few hundred node graphs [7], our approach—being based on BP—easily provides fast estimates for graphs with $100,000$ nodes. We validate the accuracy of our approach by also providing a fairly non-trivial exact counting method for clusters, utilizing advanced knowledge compilation techniques. Our approach works with the factor graph representation of the graph coloring problem. Yedidia et al. [12] showed that if one can write the so-called "partition function", $Z$, for a quantity of interest in a factor graph with non-negative weights, then there is a fairly mechanical variational method derivation that yields belief propagation equations for estimating $Z$. Under certain assumptions, we derive a partition function style quantity, $Z_{(-1)}$, to count the number of clusters. We then use the variational method to obtain BP equations for estimating $Z_{(-1)}$. Our experiments with random graph coloring problems show that $Z_{(-1)}$ itself is an extremely accurate estimate of the number of clusters, and so is its approximation, $Z_{\mathrm{BP}(-1)}$, obtained from our BP equations.

## 2 Preliminaries

The graph coloring problem can be expressed in the form of a *factor graph*, a bipartite graph with two kinds of nodes. The *variable nodes*, $\vec{x} = (x_1, \ldots, x_n)$, represent the variables in the problem ($n$ vertices to be colored) with their discrete domain $Dom = \{c_1, \ldots, c_k\}$ ($k$ colors). The *factor nodes*, $\alpha, \ldots$, with associated factor functions $f_\alpha, \ldots$, represent the constrains of the problem (no two adjacent vertices have the same color). Each factor function is a Boolean function with arguments $\vec{x}_\alpha$ (a subset of variables from $\vec{x}$) and range $\{0, 1\}$, and evaluates to 1 if and only if (iff) the associated constraint is satisfied. An edge connects a variable $x_i$ with factor $f_\alpha$ iff the variable appears in the constraint represented by the factor node, which we denote by $i \in \alpha$. In the graph coloring problem, each factor function has exactly two variables.

In the factor representation, each variable assignment $\vec{x}$ is thought of as having a weight equal to the product of the values that all factors evaluate to. We denote this product by $F(\vec{x}) := \prod_\alpha f_\alpha(\vec{x}_\alpha)$. In our case, the weight of an assignment $\vec{x}$ is 1 if all of the factors have value of 1, and 0 otherwise. The assignments with weight 1 correspond precisely to legal colorings, or *solutions* to the problem. The number of solutions can thus be expressed as the weighted sum across all possible assignments. We denote this quantity by $Z$, the so-called *partition function*:

$$Z := \sum_{\vec{x} \in Dom^n} F(\vec{x}) = \sum_{\vec{x} \in Dom^n} \prod_\alpha f_\alpha(\vec{x}_\alpha) \tag{1}$$

We define the *solution space* of a graph coloring problem to be the set of all its legal colorings. Two legal colorings (or solutions) are called *neighbors* if they differ in the color of one vertex.

**Definition 1** (Solution Cluster). A set of solutions $\mathcal{C} \subseteq \mathcal{S}$ of a solution space $\mathcal{S}$ is a *cluster* if it is a maximal subset such that any two solutions in $\mathcal{C}$ can be connected by a sequence from $\mathcal{C}$ where consecutive solutions are neighbors.

In other words, clusters are connected components of the "solution graph" which has solutions as nodes and an edge between two solutions if they differ in the value of exactly one variable.

# 3 A Partition Function Style Expression for Counting Clusters

In this section we consider a method for estimating the number of solution clusters of a graph coloring problem. We briefly describe the concepts here; a more in-depth treatment, including formal results, may be found in [8]. First let us extend the definition of the function $F$ so that it may be evaluated on an extended domain $DomExt := \mathcal{P}(\{c_1, \ldots, c_k\}) \setminus \emptyset$ where $c_1, \ldots, c_k$ are the $k$ domain values (colors) of each of the problem variables, and $\mathcal{P}$ is the power set operator (so $|DomExt| = 2^k - 1$). Each generalized assignment $\vec{y} \in DomExt^n$ thus associates a (non-empty) set of values with each original variable, defining a *hypercube* in the search space for $F$. We generalize $F$ and $f_\alpha$ to this extended domain in the natural way, $F'(\vec{y}) := \prod_{\vec{x} \in \vec{y}} F(\vec{x})$, and $f'_\alpha(\vec{y}_\alpha) := \prod_{\vec{x}_\alpha \in \vec{y}_\alpha} f_\alpha(\vec{x}_\alpha)$, where the relation $\in$ is applied point-wise, as will be the case with any relational operators used on vectors in this text. This means that $F'$ evaluates to 1 on a hypercube iff $F$ evaluates to 1 on all points within that hypercube.

Let us first assume that the solution space we work with decomposes into a set of *separated hypercubes*, so clusters correspond exactly to the hypercubes; by separated hypercubes, we mean that points in one hypercube differ from points in others in at least two values. E.g., $\vec{y}_1 = (\{c_1\}, \{c_1\}, \{c_1\})$ and $\vec{y}_2 = (\{c_2\}, \{c_3\}, \{c_1, c_2\})$ are separated hypercubes in three dimensions. This allows us to develop a surprisingly simple expression for counting the number of clusters, and we will later see that the same expression applies with high precision also to solution spaces of much more complex instances of graph coloring problems. Consider the indicator function $\chi(\vec{y})$ for the property that $\vec{y} \in DomExt^n$ is a *maximal* solution hypercube contained in the solution space:

$$\chi(\vec{y}) := \underbrace{F'(\vec{y})}_{\vec{y} \text{ is legal}} \cdot \underbrace{\prod_i \prod_{v_i \notin y_i} \left( 1 - F'(\vec{y}[y_i \leftarrow y_i \cup \{v_i\}]) \right)}_{\text{no point-wise generalization is legal}}$$

Here $\vec{y}[y_i \leftarrow y'_i]$ denotes the substitution of $y'_i$ into $y_i$ in $\vec{y}$. Note that if the solution clusters are in fact hypercubes, then variable values that can be "extended" independently can also be extended all at once, that is, $F'(\vec{y}[y_i \leftarrow y_i \cup \{v_i\}]) = 1$ and $F'(\vec{y}[y_j \leftarrow y_j \cup \{v_j\}]) = 1$ implies $F(\vec{y}[y_i \leftarrow y_i \cup \{v_i\}, y_j \leftarrow y_j \cup \{v_j\}]) = 1$. Moreover, any $F'(\vec{y}[y_i \leftarrow y_i \cup \{v_i\}])$ implies $F(\vec{y})$. Using these observations, $\chi(\vec{y})$ can be reformulated by factoring out the product as follows. Here $\#_o(\vec{y})$ denotes the number of odd-size elements of $\vec{y}$, and $\#_e(\vec{y})$ the number of even-size ones.

$$\chi(\vec{y}) = F'(\vec{y}) \left( \sum_{\vec{y}' \in (\mathcal{P}(Dom))^n \setminus \vec{y}} (-1)^{\#_o(\vec{y}')} \underbrace{\prod_i \prod_{v_i \in y'_i} F'(\vec{y}[y_i \leftarrow y_i \cup \{v_i\}])}_{= F'(\vec{y} \cup \vec{y}') \text{ by hypercube assumption}} \right)$$

$$\overset{\vec{z} := \vec{y} \cup \vec{y}'}{=} \sum_{\vec{z} \supseteq \vec{y}} (-1)^{\#_o(\vec{z} \setminus \vec{y})} F'(\vec{z}) = (-1)^{\#_e(\vec{y})} \sum_{\vec{z} \supseteq \vec{y}} (-1)^{\#_e(\vec{z})} F'(\vec{z})$$

Finally, to count the number of maximal hypercubes fitting into the set of solutions, we sum the indicator function $\chi(\vec{y})$ across all vectors $\vec{y} \in DomExt^n$:

$$\sum_{\vec{y}} \chi(\vec{y}) = \sum_{\vec{y}} (-1)^{\#_e(\vec{y})} \sum_{\vec{z} \supseteq \vec{y}} (-1)^{\#_e(\vec{z})} F'(\vec{z}) = \sum_{\vec{z}} (-1)^{\#_e(\vec{z})} F'(\vec{z}) \left( \sum_{\emptyset \notin \vec{y} \subseteq \vec{z}} (-1)^{\#_e(\vec{y})} \right)$$

$$= \sum_{\vec{z}} (-1)^{\#_e(\vec{z})} F'(\vec{z}) \left( \prod_i \underbrace{\sum_{\emptyset \notin \vec{y}_i \subseteq z_i} (-1)^{\delta_e(y_i)}}_{=1} \right) = \sum_{\vec{z}} (-1)^{\#_e(\vec{z})} F'(\vec{z})$$

The expression above is important for our study, and we denote it by $Z_{(-1)}$:

$$Z_{(-1)} := \sum_{\vec{z} \in DomExt^n} (-1)^{\#_e(\vec{z})} F'(\vec{z}) = \sum_{\vec{y} \in DomExt^n} (-1)^{\#_e(\vec{y})} \prod_\alpha f'_\alpha(\vec{y}_\alpha) \tag{2}$$

The notation $Z_{(-1)}$ is chosen to emphasize its relatedness to the partition function (1) denoted by $Z$, and indeed the two expressions differ only in the $(-1)$ term. It is easily seen that if the solution space consists of a set of separated hypercubes, then $Z_{(-1)}$ exactly captures the number of clusters (each separated hypercube is a cluster). Surprisingly, this number is remarkably accurate even for random coloring problems as we will see in Section 6, Figure 1.

# 4 Exact Computation of the Number of Clusters and $Z_{(-1)}$

Obtaining the exact number of clusters for reasonable size problems is crucial for evaluating our proposed approach based on $Z_{(-1)}$ and the corresponding BP equations to follow in Section 5. A naïve way is to explicitly enumerate all solutions, compute their pairwise Hamming distances, and infer the cluster structure. Not surprisingly, this method does not scale well because the number of solutions typically grows exponentially as the number of variables of the graph coloring problems increases. We discuss here a much more scalable approach that uses two advanced techniques to this effect: disjunctive negation normal form (DNNF) and binary decision diagrams (BDDs). Our method scales to graph coloring problems with a few hundred variables (see experimental results) for computing both the exact number of clusters and the exact value of $Z_{(-1)}$.

Both DNNF [6] and BDD [4] are graph based data structures that have proven to be very effective in "knowledge compilation", i.e., in converting a 0-1 function $F$ into a (potentially exponentially long, but often reasonably sized) standard form from which various interesting properties of $F$ can be inferred easily, often in linear time in the size of the DNNF formula or BDD. For our purposes, we use DNNF to succinctly represent all solutions of $F$ and a set of BDDs to represent solution clusters that we create as we traverse the DNNF representation. The only relevant details for us of these two representations are the following: (1) DNNF is represented as an acyclic directed graph with variables and their negations at the leaves and two kinds of internal nodes, "or" and "and"; "or" nodes split the set of solutions such that they differ in the value of the variable labeling the node but otherwise have identical variables; "and" nodes partition the space into disjoint sets of variables; (2) BDDs represent arbitrary sets of solutions and support efficient intersection and projection (onto a subset of variables) operations on these sets.

We use the compiler `c2d` [5] to obtain the DNNF form for $F$. Since `c2d` works on Boolean formulas and our $F$ often has non-Boolean domains, we first convert $F$ to a Boolean function $F'$ using a unary encoding, i.e., by replacing each variable $x_i$ of $F$ with domain size $t$ with $t$ Boolean variables $x'_{i,j}, 1 \leq j \leq t$, respecting the semantics: $x_i = j$ iff $x_{i,j} = 1$. In order to ensure that $F$ and $F'$ have similar cluster structure of solutions, we relax the usual condition that only one of $x_{i,1}, \ldots, x_{i,t}$ may be 1, thus effectively allowing the original $x_i$ to take multiple values simultaneously. This yields a *generalized function*: the domains of the variables of $F'$ correspond to the power sets of the domains of the respective variables of $F$. This generalization has the following useful property: if two solutions $\vec{x}^{(1)}$ and $\vec{x}^{(2)}$ are neighbors in the solution space of $F$, then the corresponding solutions $\vec{x}'^{(1)}$ and $\vec{x}'^{(2)}$ are in the same cluster in the solution space of $F'$.

**Computing the number of clusters.** Given $F'$, we run `c2d` on it to obtain an implicit representation of all solutions as a DNNF formula $F''$. Next, we traverse $F''$ from the leaf nodes up, creating clusters as we go along. Specifically, with each node $U$ of $F''$, we associate a set $S_U$ of BDDs, one for each cluster in the sub-formula contained under $U$. The set of BDDs for the root node of $F''$ then corresponds precisely to the set of solution clusters of $F'$, and thus of $F$. These BDDs are computed as follows. If $U$ is a leaf node of $F''$, it represents a Boolean variable or its negation and $S_U$ consists of the single one-node BDD corresponding to this Boolean literal. If $U$ is an internal node of $F''$ labeled with the variable $x_U$ and with children $L$ and $R$, the set of BDDs $S_U$ is computed as follows. If $U$ is an "or" node, then we consider the union $S_L \cup S_R$ of the two sets of BDDs and merge any two of these BDDs if they are adjacent, i.e., have two solutions that are neighbors in the solution space (since the DNNF form guarantees that the BDDs in $S_L$ and $S_R$ already must differ in the value of the variable $x_U$ labeling $U$, the adjacency check is equivalent to testing whether the two BDDs, with $x_U$ projected out, have a solution in common; this is a straightforward projection and intersection operation for BDDs); in the worst case, this leads to $|S_L| + |S_R|$ cluster BDDs in $S_U$. Similarly, if $U$ is an "and" node, then $S_U$ is constructed by considering the cross product $\{b_L \text{ and } b_R \mid b_L \in S_L, b_R \in S_R\}$ of the two sets of BDDs and merging adjacent resulting BDDs as before; in the worst case, this leads to $|S_L| \cdot |S_R|$ cluster BDDs in $S_U$.

**Evaluating $Z_{(-1)}$.** The exact value of $Z_{(-1)}$ on $F'$ can also be evaluated easily once we have the DNNF representation $F''$. In fact, as is reflected in our experimental results, evaluation of $Z_{(-1)}$ is a much more scalable process than counting clusters because it requires a simple traversal of $F''$ without the need for maintaining BDDs. With each node $U$ of $F''$, we associate a value $V_U$ which equals precisely the difference between the number of solutions below $U$ with an even number of positive literals and those with an odd number of positive literals; $Z_{(-1)}$ then equals $(-1)^N$

times the value thus associated with the root node of $F''$. These values are computed bottom-up as follows. If $U$ is a leaf node labeled with a positive (or negative) literal, then $V_U = -1$ (or 1, resp.). If $U$ is an "or" node with children $L$ and $R$, then $V_U = V_L + V_R$. This works because $L$ and $R$ have identical variables. Finally, if $U$ is an "and" node with children $L$ and $R$, then $V_U = V_L V_R$. This last computation works because $L$ and $R$ are on disjoint sets of variables and because of the following observation. Suppose $L$ has $V_L^e$ solutions with an even number of positive literals and $V_L^o$ solutions with an odd number of positive literals; similarly for $R$. Then $V_U = (V_L^e V_R^e + V_L^o V_R^o) - (V_L^e V_R^o + V_L^o V_R^e) = (V_L^e - V_L^o)(V_R^e - V_R^o) = V_L V_R$.

## 5 Belief Propagation Inference for Clusters

We present a version of the Belief Propagation algorithm that allows us to deal with the alternating signs of $Z_{(-1)}$. The derivation follows closely the one given by Yedidia et al. [12] for standard BP, i.e., we will write equations for a stationary point of KL divergence of two sequences (not necessarily probability distributions in our case). Since the $Z_{(-1)}$ expression involves both positive and negative terms, we must appropriately generalize some of the steps.

Given a function $p(\vec{y})$ (the *target* function, with real numbers as its range) on $DomExt^n$ that is known up to a normalization constant but with unknown marginal sums, we seek a function $b(\vec{y})$ (the *trial* function) to approximate $p(\vec{y})$, such that $b$'s marginal sums are known. The target function $p(\vec{y})$ is defined as $p(\vec{y}) := \frac{1}{Z_{(-1)}}(-1)^{\#_e(\vec{y})} \prod_\alpha f'_\alpha(\vec{y}_\alpha)$. We adopt previously used notation [12]: $\vec{y}_\alpha$ are values in $\vec{y}$ of variables that appear in factor (i.e. vertex) $f'_\alpha$; $\vec{y}_{-i}$ are values of all variables in $\vec{y}$ except $y_i$. The marginal sums can be extended in a similar way to allow for any number of variables fixed in $\vec{y}$, specified by the subscript. When convenient, we treat the symbol $\alpha$ as a set of indices of variables in $f'_\alpha$, to be able to index them. We begin by listing the assumptions used in the derivation, both the ones that are used in the "standard" BP, and two additional ones needed for the generalization. An assumption on $b(\vec{y})$ is *legitimate* if the corresponding condition holds for $p(\vec{y})$.

**Assumptions:** The *standard assumptions,* present in the derivation of standard BP [12], are:

- Marginalization: $b_i(y_i) = \sum_{\vec{y}_{-i}} b(\vec{y})$ and $b_\alpha(\vec{y}_\alpha) = \sum_{\vec{y}_{-\alpha}} b(\vec{y})$. This condition is legitimate, but cannot be enforced with a polynomial number of constraints. Moreover, it might happen that the solution found by BP does not satisfy it, which is a known problem with BP [10].

- Normalization: $\sum_{y_i} b_i(y_i) = \sum_{\vec{y}_\alpha} b_\alpha(\vec{y}_\alpha) = 1$. This is legitimate and explicitly enforced.

- Consistency: $\forall \alpha, i \in \alpha, y_i : b_i(y_i) = \sum_{\vec{y}_{\alpha \setminus i}} b_\alpha(\vec{y}_\alpha)$. This is legitimate and explicitly enforced.

- Tree-like decomposition: says that the weights $b(\vec{y})$ of each configuration can be obtained from the marginal sums as follows ($d_i$ is the degree of the variable node $y_i$ in the factor graph): $|b(\vec{y})| = \frac{\prod_\alpha |b_\alpha(\vec{y}_\alpha)|}{\prod_i |b_i(y_i)|^{d_i - 1}}$. (The standard assumption is without the absolute values.) This assumption is not legitimate, and it is built-in, i.e., it is used in the derivation of the BP equations.

To appropriately handle the signs of $b(\vec{y})$ and $p(\vec{y})$, we have two *additional assumptions*. These are necessary for the BP derivation applicable to $Z_{(-1)}$, but not for the standard BP equations.

- Sign-correspondence: For all configurations $\vec{y}$, $b(\vec{y})$ and $p(\vec{y})$ have the same sign (zero, being a singular case, is treated as having a positive sign). This is a built-in assumption and legitimate.

- Sign-alternation: $b_i(y_i)$ is negative iff $|y_i|$ is even, and $b_\alpha(\vec{y}_\alpha)$ is negative iff $\#_e(\vec{y}_\alpha)$ is odd. This is also a built-in assumption, but not necessarily legitimate; whether or not it is legitimate depends on the structure of the solution space of a particular problem.

The Sign-alternation assumption can be viewed as an application of the inclusion-exclusion principle, and is easy to illustrate on a graph coloring problem with only two colors. In this case, if $F'(\vec{y}) = 1$, then $y_i = \{c_1\}$ means that $y_i$ can have color 1, $y_i = \{c_2\}$ that $y_i$ can have color 2, and $y_i = \{c_1, c_2\}$ that $y_i$ can have *both* colors. The third event is included in the first two, and its probability must thus appear with a negative sign if the sum of probabilities is to be 1.

**Kullback-Leibler divergence:** The KL-divergence is traditionally defined for probability distributions, for sequences of non-negative terms in particular. We need a more general measure, as our sequences $p(\vec{y})$ and $b(\vec{y})$ have alternating signs. But using the Sign-correspondence assumption, we observe that the usual definition of KL-divergence is still applicable, since the term in the logarithm

is non-negative: $D(b \parallel p) := \sum_{\vec{y} \in DomExt^n} b(\vec{y}) \log \frac{b(\vec{y})}{p(\vec{y})} = \sum_{\vec{y} \in DomExt^n} b(\vec{y}) \log \frac{|b(\vec{y})|}{|p(\vec{y})|}$ . Moreover, the following Lemma shows that the two properties of KL-divergence that make it suitable for distance-minimization are still valid.

**Lemma 1.** *Let $b(.)$ and $p(.)$ be (possibly negative) weight functions on the same domain $\mathcal{D}$, with the property that they agree on signs for all states (i.e., $\forall \vec{y} \in \mathcal{D} : sign(b(\vec{y})) = sign(p(\vec{y}))$), and that they sum to the same constant (i.e., $\sum_{\vec{y}} b(\vec{y}) = \sum_{\vec{y}} p(\vec{y}) = c$). Then the KL-divergence $D(b \parallel p)$ satisfies $D(b \parallel p) \geq 0$ and $D(b \parallel p) = 0 \Leftrightarrow b \equiv p$.*

The proof is essentially identical to the equivalent statement made about KL-divergence of probability distributions. We omit it here for lack of space.

**Minimizing $D(b \parallel p)$:** We write $p(\vec{y}) = sign(p(\vec{y})) \cdot |p(\vec{y})|$, and analogously for $b(\vec{y})$. This allows us to isolate the signs, and the minimization follows exactly the steps of standard BP derivation, namely we write a set of equations characterizing stationary points of $D(b \parallel p)$. At the end, using the Sign-alternation assumption, we are able to implant the signs back.

**BP equations:** The resulting modified BP updates (denoted $BP_{(-1)}$) are, for $y_i \in DomExt$:

$$n_{i \to \alpha}(y_i) \quad = \quad \prod_{\beta \ni i \setminus \alpha} m_{\beta \to i}(y_i) \qquad (3)$$

$$m_{\alpha \to i}(y_i) \quad \propto \quad \sum_{\vec{y}_{\alpha \setminus i} \in DomExt^{|\alpha|-1}} f'_\alpha(\vec{y}_\alpha) \prod_{j \in \alpha \setminus i} (-1)^{\delta(|y_j| \text{ is even})} n_{j \to \alpha}(y_j) \qquad (4)$$

(Almost equivalent to standard BP, except for the $(-1)$ term.) One would iterate these equations from a suitable starting point to find a fixed point, and then obtain the beliefs $b_i(y_i)$ and $b_\alpha(\vec{y}_\alpha)$ (i.e., estimates of marginal sums) using the Sign-alternation assumption and the standard BP relations:

$$b_i(y_i) \propto (-1)^{\delta(|y_i| \text{ is even})} \prod_{\alpha \ni i} m_{\alpha \to i}(y_i) \qquad b_\alpha(\vec{y}_\alpha) \propto (-1)^{\#_e(\vec{y}_\alpha)} f'_\alpha(\vec{y}_\alpha) \prod_{i \in \alpha} n_{i \to \alpha}(y_i) \quad (5)$$

To approximately count the number of clusters in large problems for which exact cluster count or exact $Z_{(-1)}$ evaluation is infeasible, we employ the generic $BP_{(-1)}$ scheme derived above. We substitute the extended factors $f'(\vec{y}_\alpha)$ into Equations (3) and (4), iterate from a random initial starting point to find a fixed point, and then use Equations (5) to compute the beliefs. The actual estimate of $Z_{(-1)}$ is obtained with the standard BP formula (with signs properly taken care of), where $d_i$ is the degree of the variable node $y_i$ in the factor graph:

$$\log Z_{BP_{(-1)}} := -\sum_\alpha \sum_{\vec{y}_\alpha} b_\alpha(\vec{y}_\alpha) \log |b_\alpha(\vec{y}_\alpha)| + \sum_i (d_i - 1) \sum_{y_i} b_i(y_i) \log |b_i(y_i)| \qquad (6)$$

## 6 Experimental Evaluation

We empirically evaluate the accuracy of our $Z_{(-1)}$ and $Z_{BP_{(-1)}}$ approximations on an ensemble of random graph 3-coloring instances. The results are discussed in this section.

**$Z_{(-1)}$ vs. the number of clusters.** The left panel of Figure 1 compares the number of clusters (on the x-axis, log-scale) with $Z_{(-1)}$ (on the y-axis, log-scale) for $2,500$ colorable random 3-COL instances on graphs with 20, 50, and 100 vertices with average vertex degree ranging between $1.0$ and $4.7$ (the threshold for 3-colorability). As can be seen, the $Z_{(-1)}$ expression captures the number of clusters almost exactly. The inaccuracies come mostly from low graph density regions; in all instances we tried with density $> 3.0$, the $Z_{(-1)}$ expression was exact. We remark that although uncolorable instances were not considered in this comparison, $Z_{(-1)} = 0 = $ num-clusters by construction.

It is worth noting that for tree-structured graphs (with more than one vertex), the $Z_{(-1)}$ expression gives 0 for any $k \geq 3$ colors although there is exactly one solution cluster. Moreover, given a disconnected graph with at least one tree component, $Z_{(-1)}$ also evaluates to 0 as it is the product of $Z_{(-1)}$ values over different components. We have thus removed all tree components from the generated graphs prior to computing $Z_{(-1)}$; tree components are easily identified and removing them does not change the number of clusters. For low graph densities, there are still some instances

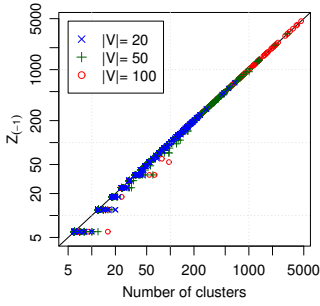
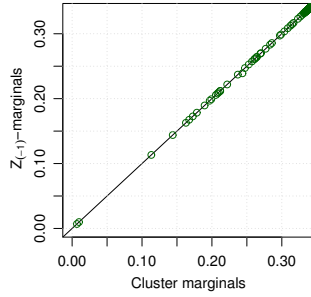
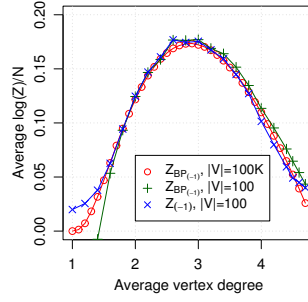

Figure 1: Left: $Z_{(-1)}$ vs. number of clusters in random 3-COL problems with $20$, $50$ and $100$ vertices, and average vertex degree between $1.0 - 4.7$. Right: cluster marginals vs. $Z_{(-1)}$-marginals for one instance of random 3-COL problem with $100$ vertices.

Figure 2: Average $Z_{\mathrm{BP}_{(-1)}}$ and $Z_{(-1)}$ for 3-COL vs. average vertex degrees for small and large random graphs.

for which $Z_{(-1)}$ evaluates to $0$; these instances are not visible in Figure 1 due to the log-log scale. In fact, all our instances with fewer than 5 clusters have $Z_{(-1)} = 0$. This is because of other substructures for which $Z_{(-1)}$ evaluates to $0$, e.g., cordless cycles of length not divisible by 3 (for $k = 3$ coloring) with attached trees. These structures, however, become rare as the density increases.

**$Z_{(-1)}$ marginals vs. clusters marginals.** For a given problem instance, we can define the *cluster marginal* of a variable $x_i$ to be the fraction of solution clusters in which $x_i$ only appears with one particular value (i.e., $x_i$ is a backbone of the cluster). Since $Z_{(-1)}$ counts well the number of clusters, it is natural to ask whether it is also possible to obtain the marginals information from it. Indeed, $Z_{(-1)}$ does provide an estimate of the cluster marginals, and we call them $Z_{(-1)}$-*marginals*. Recall that the semantics of factors in the extended domain is such that a variable can assume a set of values only if *every* value in the set yields a solution to the problem. This extends to the $Z_{(-1)}$ estimate of the number of clusters, and one can therefore use the principle of inclusion-exclusion to compute the number of clusters where a variable can only assume one particular value. The definition of $Z_{(-1)}$ conveniently provides for correct signs, and the number of clusters where $x_i$ is fixed to $v_i$ is thus estimated by $\sum_{y_i \ni v_i} Z_{(-1)}(y_i)$, where $Z_{(-1)}(y_i)$ is the marginal sum of $Z_{(-1)}$. The $Z_{(-1)}$-marginal is obtained by dividing this quantity by $Z_{(-1)}$.

The right panel of Figure 1 shows the results on one random 3-COL problem with $100$ vertices. The plot shows cluster marginals and $Z_{(-1)}$-marginals for one color; the points correspond to individual variables. The $Z_{(-1)}$-marginals are close to perfect. This is a typical situation, although it is important to mention that $Z_{(-1)}$-marginals are not always correct, or even non-negative. They are merely an estimate of the true cluster marginals, and how well they work depends on the solution space structure at hand. They are exact if the solution space decomposes into separated hypercubes and, as the figure shows, remarkably accurate also for random coloring instances.

**The number of clusters vs. $Z_{\mathrm{BP}_{(-1)}}$.** Figure 3 depicts a comparison between $Z_{\mathrm{BP}_{(-1)}}$ and $Z_{(-1)}$ for the 3-COL problem on colorable random graphs of various sizes and graph densities. It compares $Z_{(-1)}$ (on the x-axis, log-scale) with $Z_{\mathrm{BP}_{(-1)}}$ (y-axis, log-scale) for $1,300$ colorable 3-COL instances on random graphs with $50$, $100$, and $200$ vertices, with average vertex degree ranging from $1.0$ to $4.7$. The plots shows that BP is quite accurate in estimating $Z_{(-1)}$ for individual instances, which in turn captures the number of clusters. Instances which are not 3-colorable are not shown, and BP in general incorrectly estimates a non-zero number of clusters for them.

**Estimates on very large graphs and for various graph densities.** Figure 2 shows similar data from a different perspective: what is shown is a rescaled average estimate of the number of clusters (y-axis) for average vertex degrees $1.0$ to $4.7$ (x-axis). The average is taken across different colorable instances of a given size, and the rescaling assumes that the number of clusters $= \exp(|V| \cdot \Sigma)$ where $\Sigma$ is a constant independent of the number of vertices [3]. The three curves show, respectively, BP's estimate for graphs with $100,000$ vertices, BP's estimate for graphs with $100$ vertices, and $Z_{(-1)}$ for the same graphs of size $100$. The averages are computed across $3,000$ instances of the small graphs, and only $10$ instances of the large ones where the instance-to-instance variability is practically non-existent. The fact that the curves nicely overlay shows that $\mathrm{BP}_{(-1)}$ computes $Z_{(-1)}$ very accurately

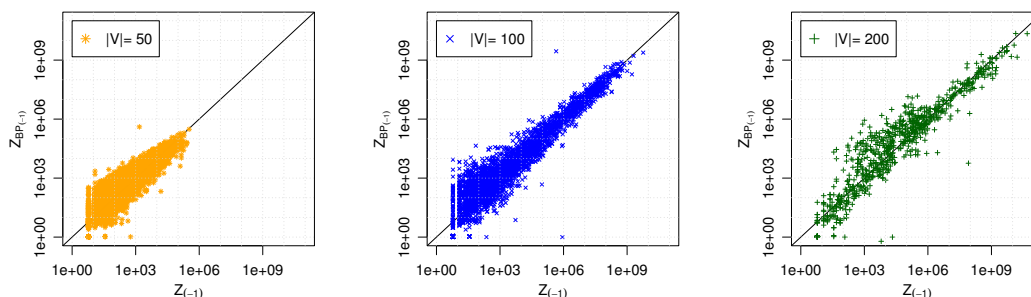

Figure 3: $Z_{\mathrm{BP}_{(-1)}}$ compared to $Z_{(-1)}$ for 3-COL problem on random graphs with $50$, $100$ and $200$ vertices and average vertex degree in the range $1.0 - 4.7$.

on average for colorable instances (where we can compare it with exact values), and that the estimate remains accurate for large problems. Note that the Survey Propagation algorithm developed by Braunstein et al. [3] also aims at computing the number of certain clusters in the solution space. However, SP counts only the number of clusters with a "typical size", and would show non-zero values in Figure 2 only for average vertex degrees between $4.42$ and $4.7$. Our algorithm counts clusters of all sizes, and is very accurate in the entire range of graph densities.

## 7    Conclusion

We discuss a purely combinatorial construction for estimating the number of solution clusters in graph coloring problems with very high accuracy. The technique uses a hypercube-based inclusion-exclusion argument coupled with solution counting, and lends itself to an application of a modified belief propagation algorithm. This way, the number of clusters in huge random graph coloring instances can be accurately and efficiently estimated. Our preliminary investigation has revealed that it is possible to use combinatorial arguments to formally prove that the cluster counts estimated by $Z_{(-1)}$ are exact on certain kinds of solution spaces (not necessarily only for graph coloring). We hope that such insights and the cluster-focused picture will lead to new techniques for solving hard combinatorial problems and for bounding solvability transitions in random problem ensembles.

## References

[1]  D. Achlioptas and F. Ricci-Tersenghi. On the solution-space geometry of random constraint satisfaction problems. In *38th STOC*, pages 130–139, Seattle, WA, May 2006.

[2]  J. Ardelius, E. Aurell, and S. Krishnamurthy. Clustering of solutions in hard satisfiability problems. *J. Statistical Mechanics*, P10012, 2007.

[3]  A. Braunstein, R. Mulet, A. Pagnani, M. Weigt, and R. Zecchina. Polynomial iterative algorithms for coloring and analyzing random graphs. *Physical Review E*, 68:036702, 2003.

[4]  R. E. Bryant. Graph-based algorithms for Boolean function manipulation. *IEEE Transactions on Computers*, 35(8):677–691, 1986.

[5]  A. Darwiche. New advances in compiling CNF into decomposable negation normal form. In *16th European Conf. on AI*, pages 328–332, Valencia, Spain, Aug. 2004.

[6]  A. Darwiche. Decomposable negation normal form. *J. ACM*, 48(4):608–647, 2001.

[7]  A. Hartmann, A. Mann, and W. Radenback. Clusters and solution landscapes for vertex-cover and SAT problems. In *Workshop on Physics of Distributed Systems*, Stockholm, Sweden, May 2008.

[8]  L. Kroc, A. Sabharwal, and B. Selman. Counting solution clusters of combinatorial problems using belief propagation, 2008. (in preparation).

[9]  F. Krzakala, A. Montanari, F. Ricci-Tersenghi, G. Semerjian, and L. Zdeborova. Gibbs states and the set of solutions of random constraint satisfaction problems. *PNAS*, 104(25):10318–10323, June 2007.

[10]  D. Mackay, J. Yedidia, W. Freeman, and Y. Weiss. A conversation about the Bethe free energy and sum-product, 2001. URL citeseer.ist.psu.edu/mackay01conversation.html.

[11]  M. Mézard, G. Parisi, and R. Zecchina. Analytic and algorithmic solution of random satisfiability problems. *Science*, 297(5582):812–815, 2002.

[12]  J. S. Yedidia, W. T. Freeman, and Y. Weiss. Constructing free-energy approximations and generalized belief propagation algorithms. *IEEE Transactions on Information Theory*, 51(7):2282–2312, 2005.
